# What Rotary Position Embedding Can Tell Us: Identifying Query and Key Weights Corresponding to Basic Syntactic or High-level Semantic Information

**Yiting Chen, Junchi Yan**‡

Dept. of CSE & School of AI & MoE Key Lab of AI, Shanghai Jiao Tong University
{sjtuucyt, yanjunchi}@sjtu.edu.cn
https://github.com/Ytchen981/RoPE_investigate

## Abstract

Transformer-based large language models (LLMs) have successfully handled various tasks. As one fundamental module in Transformers, position encoding encodes the positional information of tokens in a sequence. Specifically, rotary position embedding (RoPE), one of the most widely used techniques, encodes the positional information by dividing the query or key value with $d$ elements into $d/2$ pairs and rotating the 2d vectors corresponding to each pair of elements. Therefore, the direction of each pair and the position-related rotation jointly determine the attention score. In this paper, we show that the direction of the 2d pair is largely affected by the angle between the corresponding weight vector pair. We theoretically show that non-orthogonal weight vector pairs lead to great attention on tokens at a certain relative position and are less sensitive to the input which may correspond to basic syntactic information. Meanwhile, the orthogonal weight vector pairs are more flexible regarding the relative position, which may correspond to high-level syntactic information. Empirical evidence supports the hypothesis that shallow layers of LLMs focus more on local syntax and deep layers focus more on high-level semantics. Furthermore, we show that LLMs fine-tuning mainly changes the pairs of weight vectors that are nearly orthogonal, i.e., the weight corresponding to high-level semantics, which enables the reduction of the number of trainable parameters during fine-tuning without sacrificing performance. We propose a method namely Angle-based Weight Masking (AWM) to reduce the fine-tuning overhead and verify the effectiveness of the proposed method on widely used Alpaca fine-tuned Llama-2.

## 1 Introduction

Large language models [36, 26, 35] have achieved impressive success and attracted considerable attention towards analyzing the Transformer structure and enhancing fine-tuning efficiency [14, 1, 18, 15]. As one fundamental module in a Transformer, position encoding encodes the positional information of tokens in a sequence. Various position encoding have been proposed including absolute position encoding [36, 8, 26], and relative position encoding [30, 27], *etc.* Among them, one of the most widely used position encodings is rotary position embedding (RoPE) [31] which is used in various popular LLMs such as Llama [35], Mistral [16], GLM [9], *etc.* Instead of applying a function to the input, RoPE applies a transformation to the query and the key of each attention layer, which provides us with a window to investigate how the LLMs utilize the position information. In

this paper, we analyze the query and the key weight matrix in the self-attention module of LLMs using RoPE and propose a simple method to reduce the computational cost of LLM fine-tuning.

As the name "rotary" indicates, RoPE divides the elements of queries and keys into pairs and rotates each pair as a 2D vector with a certain angle determined by the position. One simple fact is that the direction of the 2D vector before rotation and the position-related rotation jointly affect the attention score. Note that the direction of the 2D vector is determined by the corresponding weight vector pairs. As illustrated in Fig. 1, a simple analysis shows that non-orthogonal weight vector pairs with large absolute cosine similarity values are less sensitive to the input and will draw greater attention to certain positions. It provides an indicator of how the model utilizes the position information. We empirically show that attention heads with large absolute cosine similarity between weight vector pairs focus more on basic syntactic information while attention heads with near-zero cosine similarity between weight vector pairs focus more on high-level semantics. In linguistics, formally, syntactic information refers to the arrangement of symbols or words according to the rules of a formal system or language [10] and semantic information pertains to the meaning and interpretation of words, phrases, and sentences [28]. In studying how deep learning models utilise syntactic and semantic information, previous works [34] empirically show that shallow layers of LLMs focus more on basic syntactic information and deep layers of LLMs focus more on high-level semantics. Our experimental results align with previous works and further support the hypothesis. In a more fine-grained perspective, we further show that within each attention head, the angles between weight vector pairs also vary.

By comparing the fine-tuned version with the pre-trained version of LLMs, we further show that the weight in the query and the key are mainly changed on nearly orthogonal weight vector pairs during fine-tuning, and the non-orthogonal weight vector pairs are barely changed. This implies that fine-tuning the query and the key in LLMs mainly changes the weights corresponding to high-level semantic information and does not change the weights corresponding to basic syntactic information in the query and key. We conjecture that it is because the pre-trained LLMs are already good enough in processing basic syntactic information and only need to be tuned on how to process high-level semantic information for downstream tasks. Therefore, we propose to fix the non-orthogonal pairs of weight vectors in the query and key of each layer in the pre-trained models to reduce the number of trainable parameters during fine-tuning. We conduct experiments on widely used models and datasets to verify the effectiveness of our method. We show that our method could effectively reduce the number of trainable parameters while maintaining or even boosting the performance of the fine-tuned model. **We summarize the contributions of this paper in the following:**

- We provide a new perspective to investigate how LLMs with RoPE utilize the positional information and theoretically show that non-orthogonal weight vector pairs divided by RoPE are less sensitive to input and draw greater attention to certain relative positions.
- We empirically show the angles between the weight vector pairs in the query and the key could serve as an indicator of whether the pair of weight vectors focuses on basic syntactic information or high-level semantics. Non-orthogonal weight vector pairs focus more on basic syntactic information while nearly orthogonal weight vector pairs focus more on high-level semantic information.
- Since RoPE divides the elements in the query or key values into pairs, we empirically show that fine-tuning LLM mainly changes the orthogonal pairs of corresponding weight vectors. Based on our findings, only orthogonal pairs of weight vectors are changed during fine-tuning, we propose a method to reduce the number of trainable parameters during LLM fine-tuning and verify its effectiveness on widely used models and benchmarks.

## 2    Preliminaries and Related Works

**Position Encoding.** After the seminar work [36], various Transformer-based position encoding methods have been proposed to incorporate position information into the function calculating query, key, and value. One typical way is adding the input with a vector depending on the position of the input vector. Learnable position embedding is introduced where the position embedding is trainable during training [36, 8, 26]. The sinusoidal positional encoding was introduced by the authors of [36] where the vector is generated by a sinusoidal function. Instead of using absolute position, another branch of work proposes relative position encoding [30, 27]. Rotary position embedding (RoPE) [31] was proposed to incorporate position information by rotating the pairs of elements in the query and the key, which was widely used in different LLMs such as Llama [35], Palm [2], Mistral [16], GLM [9], *etc.* Recent works on long-context Transformers also show the advantage of RoPE on input

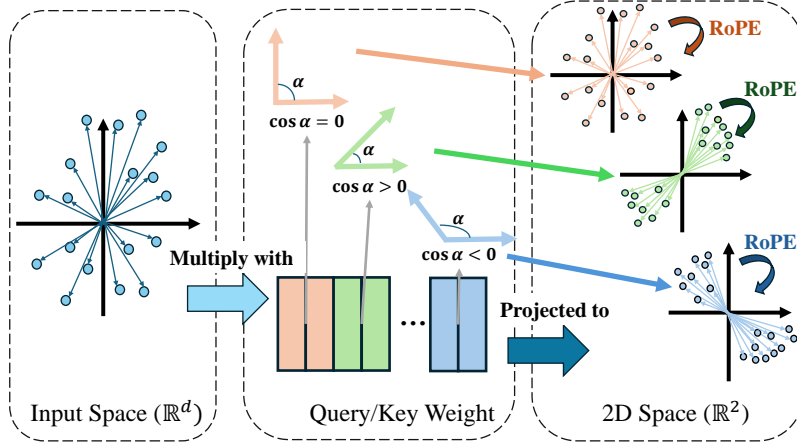

Figure 1: Illustration of how the angle between weight vector pairs in the query or the key affects RoPE. The larger absolute cosine similarity $|\cos\alpha|$, the direction of the projected 2D vector is more fixed which leads to high attention on certain relative positions regardless of the input.

length extrapolation [32, 24]. In this paper, we provide a new weight vector angle perspective to investigate how LLMs with RoPE utilize the positional information and empirically show that we can identify weights corresponding to processing basic syntactic information or high-level semantic information. Please refer to Sec. 3 for more details.

**Parameter-Efficient Fine-Tuning.** Different from full fine-tuning that fine-tunes all the parameters, parameter-efficient fine-tuning (PEFT) methods aim at reducing the number of parameters fine-tuned to reduce the computational cost and prevent overfitting on fine-tuning data [25]. Various PEFT methods have been proposed such as adapter tuning [14], prefix tuning [1], and prompt tuning [18]. Adapter tuning proposes adding a small module to each layer of the pre-trained model. Prefix tuning and prompt tuning propose adding additional tunable prefix tokens to the input. Notably, LoRA [15] was proposed to use a low-rank matrix to approximate parameter updates.

**Other Methods to Reduce Computational Cost of LLMs.** Besides the PEFT methods, many other methods are proposed to accelerate the LLMs. Quantization [5, 7] is a common technique to reduce the memory footprint of LLMs. [6] propose double quantization to further extend the application of quantization from inference to fine-tuning. Another common technique is pruning where parameters or modules are pruned to reduce the computational cost [17, 13]. Recently, [12] shows that deep layers of LLMs can be pruned without degrading much performance. Orthogonal to these previous methods in reducing the computational cost of LLMs, we provide a method to identify weights in the query and the key that does not require updating during fine-tuning. Refer to Sec. 4 for more details.

**Definition of RoPE.** For an input sequence $\mathbb{S}_N = \{\mathbf{x}_m\}_{m=1}^N$ where $\mathbf{x}_m \in \mathbb{R}^d$ is the input regarding the $m$-th token. The self-attention with RoPE [31] generates the query and the key with

$$\begin{aligned} \mathbf{q}_m &= f_q(x_m, m) = R_{\Theta,m}^d W_q \mathbf{x}_m, \\ \mathbf{k}_m &= f_k(x_m, m) = R_{\Theta,m}^d W_k \mathbf{x}_m, \end{aligned} \tag{1}$$

where $\mathbf{q}_m$ and $\mathbf{k}_m$ is the query and key of the $m$-th token, the $W_q$ and $W_k$ is the weight matrix and $R_{\Theta,i}^d$ is the rotary matrix defined as following

$$R_{\Theta,m}^d = \begin{pmatrix} \cos m\theta_1 & -\sin m\theta_1 & 0 & 0 & \cdots & 0 & 0 \\ \sin m\theta_1 & \cos m\theta_1 & 0 & 0 & \cdots & 0 & 0 \\ 0 & 0 & \cos m\theta_2 & -\sin m\theta_2 & \cdots & 0 & 0 \\ 0 & 0 & \sin m\theta_2 & \cos m\theta_2 & \cdots & 0 & 0 \\ \vdots & \vdots & \vdots & \vdots & \ddots & \vdots & \vdots \\ 0 & 0 & 0 & 0 & \cdots & \cos m\theta_{d/2} & -\sin m\theta_{d/2} \\ 0 & 0 & 0 & 0 & \cdots & \sin m\theta_{d/2} & \cos m\theta_{d/2} \end{pmatrix}. \tag{2}$$

The $\Theta$ is predefined as $\Theta = \{\theta_i = 10000^{-2(i-1)/d}, i \in [1, 2, \cdots, d/2]\}$.

# 3 Investigating the Angle between Weight Vector Pairs in RoPE

In this section, we first provide a simple analysis regarding how the angle between the pair of weight vectors will affect the RoPE and then provide empirical results regarding how the LLM utilizes the positional information with different attention heads and across layers.

## 3.1 A Simple Analysis Regarding the Angle between Weight Vector Pairs

Let $\mathbf{w}_q^{i\top}$ and $\mathbf{w}_k^{i\top}$ denote the $i$-th row vector of $W_q$ and $W_k$ such that $W_q = (\mathbf{w}_q^1, \mathbf{w}_q^2, \cdots, \mathbf{w}_q^d)^\top$ and $W_k = (\mathbf{w}_k^1, \mathbf{w}_k^2, \cdots, \mathbf{w}_k^d)^\top$. As shown in Eq. 1, RoPE rotates each pair $(\mathbf{w}_q^{(2i-1)\top}\mathbf{x}_m, \mathbf{w}_q^{(2i)\top}\mathbf{x}_m)$ and $(\mathbf{w}_k^{(2i-1)\top}\mathbf{x}_m, \mathbf{w}_k^{(2i)\top}\mathbf{x}_m)$ with $m\theta_i$. As we take the inner product of query and key, the direction of 2D pairs and the position-related rotation jointly determine the attention weight. According to the definition in Eq. 1 we have

$$\mathbf{q}_n^\top \mathbf{k}_m = \sum_{i=1}^{d/2} \|(\mathbf{w}_q^{(2i-1)\top}\mathbf{x}_n, \mathbf{w}_q^{(2i)\top}\mathbf{x}_n)\| \cdot \|(\mathbf{w}_k^{(2i-1)\top}\mathbf{x}_m, \mathbf{w}_k^{(2i)\top}\mathbf{x}_m)\| \cdot \cos\left(\gamma_{n,q}^i - \gamma_{m,k}^i + (n-m)\theta_i\right).$$
(3)

Where $\gamma_{n,q}^i$ corresponds to the direction of 2D vector $(\mathbf{w}_q^{(2i-1)\top}\mathbf{x}_n, \mathbf{w}_q^{(2i)\top}\mathbf{x}_n)$ and $\gamma_{m,k}^i$ corresponds to the direction of 2D vector $(\mathbf{w}_k^{(2i-1)\top}\mathbf{x}_m, \mathbf{w}_k^{(2i)\top}\mathbf{x}_m)$.

Let us consider the angle $\gamma_{n,q}^i$ and $\gamma_{m,k}^i$. Since the analysis for the query and the key are the same, we use $\{q, k\}$ to represent either the query or the key. Let $\mathbf{x}_{m,\{q,k\}}^i$ denotes the projection of $\mathbf{x}_m$ onto the subspace held by $\mathbf{w}_{\{q,k\}}^{2i-1}$ and $\mathbf{w}_{\{q,k\}}^{2i}$. Suppose the angle between $\mathbf{w}_{\{q,k\}}^{2i-1}$ and $\mathbf{w}_{\{q,k\}}^{2i}$ is $\alpha_{\{q,k\}}^i$ and the angle between $\mathbf{w}_{\{q,k\}}^{2i-1}$ and $\mathbf{x}_{m,\{q,k\}}^i$ is $\beta_{m,\{q,k\}}^i$, we have

$$\tan\gamma_{m,\{q,k\}}^i = \frac{\mathbf{w}_q^{(2i)\top}\mathbf{x}_m}{\mathbf{w}_q^{(2i-1)\top}\mathbf{x}_m} = \frac{\|\mathbf{x}_{m,\{q,k\}}^i\|\|\mathbf{w}_{\{q,k\}}^{2i}\|\cos(\beta_{m,\{q,k\}}^i - \alpha_{\{q,k\}}^i)}{\|\mathbf{x}_{m,\{q,k\}}^i\|\|\mathbf{w}_{\{q,k\}}^{2i-1}\|\cos(\beta_{m,\{q,k\}}^i)}$$
(4)

Derive Eq. 4, we get

$$\gamma_{m,\{q,k\}}^i = \arctan\left(\frac{\|\mathbf{w}_{\{q,k\}}^{2i}\|}{\|\mathbf{w}_{\{q,k\}}^{2i-1}\|} \cdot \left(\cos\alpha_{\{q,k\}}^i + \sin\alpha_{\{q,k\}}^i \tan\beta_{m,\{q,k\}}^i\right)\right).$$
(5)

As shown in Eq. 5, the larger $|\sin\alpha_{\{q,k\}}^i|$ the larger impact $\beta_{m,\{q,k\}}^i$ would have on $\gamma_{m,\{q,k\}}^i$. It indicates that if the $|\sin\alpha_{\{q,k\}}^i|$ is small and the $|\cos\alpha_{\{q,k\}}^i|$ is large for both the query and the key, the attention weight would be less sensitive to the input and draw greater attention to certain relative positions[1] since the directions of the projected 2-d vector $(\mathbf{w}_{\{q,k\}}^{(2i-1)\top}\mathbf{x}_m, \mathbf{w}_{\{q,k\}}^{(2i)\top}\mathbf{x}_m)$ of every token are close. In an extreme condition, when the two weight vectors are in the same direction or opposite direction, $\sin\alpha_{\{q,k\}}^i = 0$, we have $\gamma_{m,\{q,k\}}^i = \arctan(\frac{\|\mathbf{w}_{\{q,k\}}^{2i}\|}{\|\mathbf{w}_{\{q,k\}}^{2i-1}\|})$, which means the direction of the 2-d vector $(\mathbf{w}_{\{q,k\}}^{(2i-1)\top}\mathbf{x}_m, \mathbf{w}_{\{q,k\}}^{(2i)\top}\mathbf{x}_m)$ is fixed for any $\mathbf{x}_m$.

Note that as vectors in high-dimensional space, due to the curse of dimensionality, these weight vector pairs are nearly orthogonal when randomly initialized. Therefore the non-orthogonal weight vector pairs are non-trivial such that the model learns non-orthogonal weight vector pairs to emphasize tokens at certain relative positions. In the following sections, we empirically show that non-orthogonal weight vector pairs widely exist in LLMs and provide a new perspective for us to investigate LLMs.

## 3.2 Attention Visualization for Different Attention Heads

To verify the conjecture in Sec. 3.1 that large absolute cosine similarity $|\cos\alpha|$ corresponds to basic syntactic information and small $|\cos\alpha|$ corresponds to high-level semantics, we visualize the

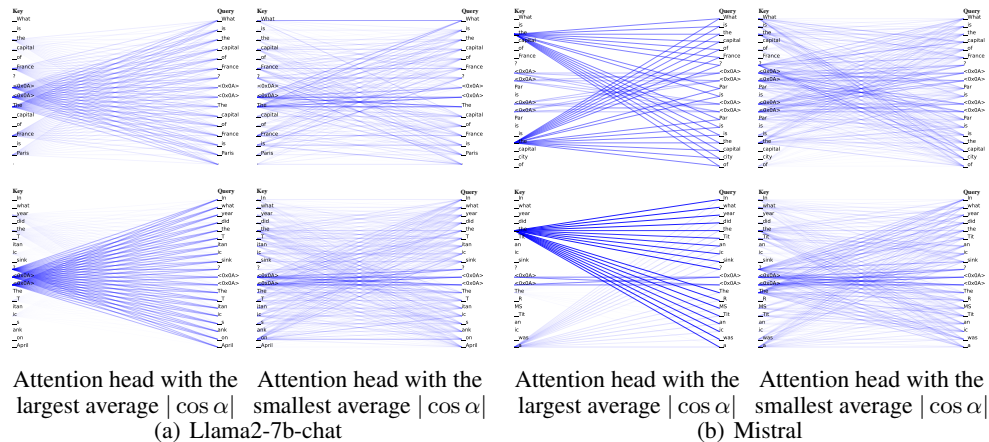

Attention head with the largest average $|\cos\alpha|$ | Attention head with the smallest average $|\cos\alpha|$ | Attention head with the largest average $|\cos\alpha|$ | Attention head with the smallest average $|\cos\alpha|$

(a) Llama2-7b-chat | (b) Mistral

Figure 2: Attention visualization of the attention heads with the largest or the smallest average absolute cosine similarity $|\cos\alpha|$ across the weight vector pairs in RoPE, where the key is on the left, and the query is on the right. The lower transparency means a higher attention score. We demonstrate the results for the first layer of Llama-2-7b-chat and Mistral-7B-Instruct-v0.2. The results empirically support that higher $|\cos\alpha|$ leads to attention on basic syntactic information and lower $|\cos\alpha|$ leads to attention on high-level semantic information. For more results of different models at different layers please refer to Appendix B.

attention of attention heads with different average absolute cosine similarity $|\cos\alpha|$ in this section. With simple questions such as "What is the capital of France?" as input, we record the attention score of different attention heads and visualize the score by setting the transparency of the lines connecting the key and the query accordingly. The lower transparency corresponds to a higher attention score. For clarity in the figure, we limit the number of tokens to 20.

As shown in Fig. 2, we present the attention visualization of the attention heads with the largest or the smallest average absolute cosine similarity in the 1st layer of Llama-2-7b-chat [35] and Mistral-7b-Instruct-v0.2 [16]. For the attention head with a large absolute cosine similarity value across the weight vector pairs (0.54 on average for the 1st layer of Llama2-7b and 0.65 on average for the 1st layer of Mistral-7b), attention is mainly on tokens of prepositions or articles that may correspond more to the basic syntactic information. Notably, attention to special tokens is high. Special tokens mainly correspond to syntactic information, such as the end of the input prompt or the start of the answer by LLMs. In contrast, for the attention head with a small absolute cosine similarity value across the weight vector pairs in RoPE (0.24 on average for the 1st layer of Llama2-7b and 0.29 on average for the 1st layer of Mistral-7b), the attention is on every token of the phrase which may correspond more to the high-level semantics. For more results at different layers or different models, please refer to Appendix. B.

### 3.3 Analyzing Weight Vector Pair Angles Across the Layers

In this section, we propose to investigate how LLMs utilize positional information from RoPE across the layers from the weight vector angle perspective. For each layer, we calculate the average absolute value of cosine similarity across all the weight vector pairs in the layer.

As shown in Fig. 3, we report the results of each layer of different LLMs. For each LLM, generally, the average absolute cosine value decreases to a very low value after the first several layers (typically 3 layers) and stays low for the rest of the layers until it is slightly increased at the last layer. This phenomenon agrees with the hypothesis that the LLMs first process the information about local syntax at the first several layers and then process the high-level semantic information [34, 3]. While the angle between weight vectors pairs in RoPE also provides a new perspective complementary to the probing tasks [34] designed to show that models like BERT [8] process the information layer-by-layer following the traditional NLP pipeline. It also explains the success of fastly training a smaller model using the first several layers of a pre-trained LLM [29] where the first several layers contain the weights responsible for processing basic syntactic information, which plays a vital role in the model.

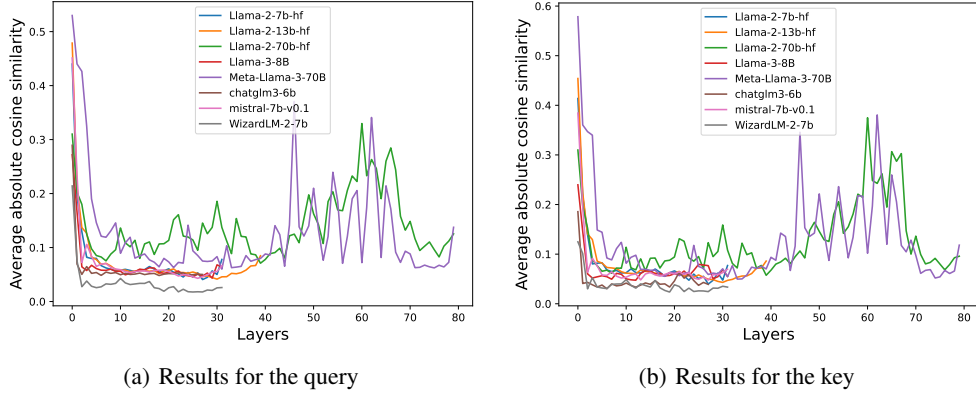

| (a) Results for the query | (b) Results for the key |

Figure 3: We report the average absolute cosine similarity of the query and the key across the layers of different LLMs. We show that, for all the LLMs we investigate, the average absolute cosine similarity drastically decreases after the first several layers and stays small until the last layer.

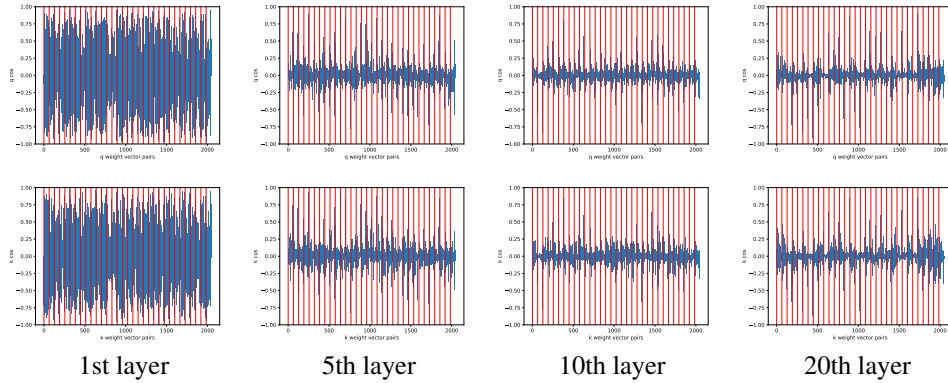

| 1st layer | 5th layer | 10th layer | 20th layer |

Figure 4: We show the cosine similarity of each weight vector pair in the query and the key of different layers of the Llama2-7b [35]. Each column corresponds to different layers. The top row is the results for the query and the bottom row is the results for the key. The attention heads are separated with vertical red lines. We show that even within one head, the cosine similarity of different weight vector pairs differs. More results on other layers and other models are in Appendix B.

The last layer generates the output in a certain format, which leads to a slight increase in the average absolute cosine value. This also agrees with the empirical results in previous works [37, 21] that the sparsity of activation changes in the middle from sparse to dense, and the layers at the middle could be pruned. [3] also shows that the layers in the middle are the mostly changed layers during fine-tuning. In Sec. 4, we further extend the result to that the orthogonal pairs of weight vectors ($\cos \alpha = 0$) are the mostly changed weights during fine-tuning and propose a method to reduce the number of trainable parameters during fine-tuning.

### 3.4 Investigating the Cosine Similarity Across Weight Vector Pairs

Beyond the coarse grain results (attention head-wise as in Sec. 3.2 and layer-wise as in Sec. 3.3), the angles between weight vector pairs could provide a more fine-grained view. In this section, we propose to investigate the distribution of cosine similarity between weight vector pairs in the query and the key. As shown in Fig. 4, we report the cosine similarity, the $\cos \alpha$, of weight vector pairs of the query and key in the first layer of Llama2-7b [35]. In the figure, we separate different attention heads by vertical red lines. The cosine similarity changes drastically across the heads or even across the pairs of weight vectors in the same head. This means that in one head, there is still a division of labor, such that some of the weight vector pairs draw more attention to certain relative positions while the other weights are more flexible. More results of different LLMs are in Appendix B.

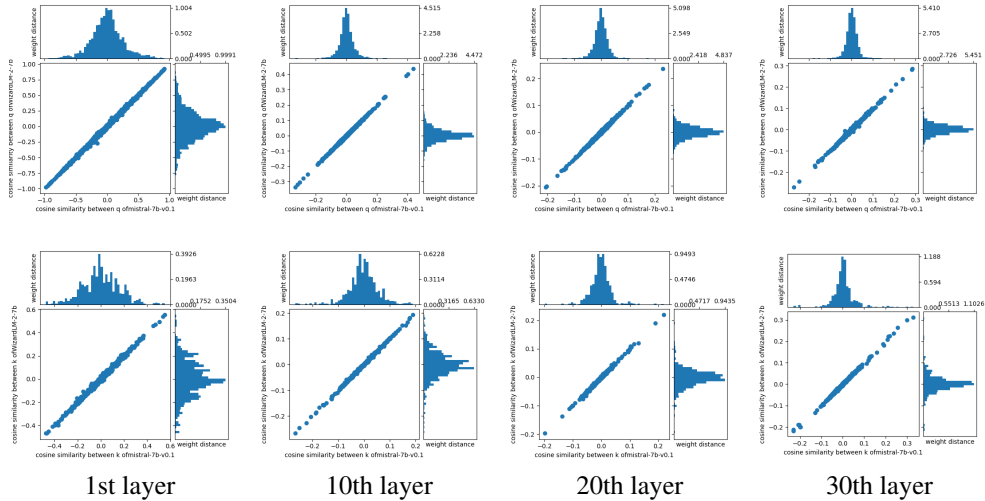

| 1st layer | 10th layer | 20th layer | 30th layer |

Figure 6: Comparison between the base model Mistral-7B [16] and fine-tuned version WizardLM-2 [38]. For each sub-figure, the scatter figure in the middle demonstrates the cosine similarity between weight vector pairs where each point corresponds to a weight vector pair; the x-axis corresponds to the results for the base model, and the y-axis corresponds to the fine-tuned model. In the histogram on the top and left, we report the average $L_2$ weight distance between the weight vectors of the pre-trained model and the fine-tuned model. The y-axis of the histogram on the top and the x-axis of the histogram on the left correspond to the $L_2$ weight distance. Generally, fine-tuning merely changes the angle between weight vector pairs. Besides the first several layers, the weight change mainly happens on weight vector pairs that are nearly orthogonal. More results are provided in Appendix B.

Still, we find that the angles between the pairs of weight vectors in query and key are highly related to each other. As shown in Fig. 5, the $\cos \alpha$ of pairs of weights of the query is positively correlated with the $\cos \alpha$ of pairs of weights of the key with Pearson correlation at $0.86$. It makes sense since it requires both the angle of the projected 2D vector of the query and the key to be nearly fixed to assure high attention on certain relative positions. Initialized to be both nearly orthogonal, we conjecture that the angles between the query and the key weight vector pairs may be changed simultaneously during training. For results of other layers and other models, refer to Appendix B.

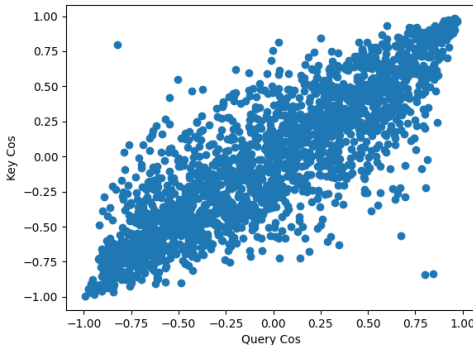

Figure 5: Correlation between cosine similarity of the query and key weight vector pair of the 1st layer of Llama2-7b. Each dot represents two corresponding weight vector pairs in query and key. The x-axis corresponds to the cosine similarity of the weight vector pair in the query and the y-axis corresponds to the cosine similarity of the weight vector pair in the key. (Pearson's r is 0.86)

# 4 Reducing the Trainable Parameters During Fine-tuning

In this section, we first show that only the weight vector pairs that are nearly orthogonal (with $\cos \alpha = 0$) are changed during training by comparing the base version and finetuned version (*e.g.* chat or instruct version) of the same LLM. We then propose an efficient and effective method, namely Angle-based Weight Masking (AWM), that fixes the non-orthogonal query and key weight vector pairs defined in RoPE to reduce the number of trainable parameters while maintaining or even boosting the performance during fine-tuning. Since our proposed method fixes parameters at a fine-grained weight row vector level, it is orthogonal to LoRA [15], the widely used parameter-efficient fine-tuning method. The experimental results verified the effectiveness of AWM.

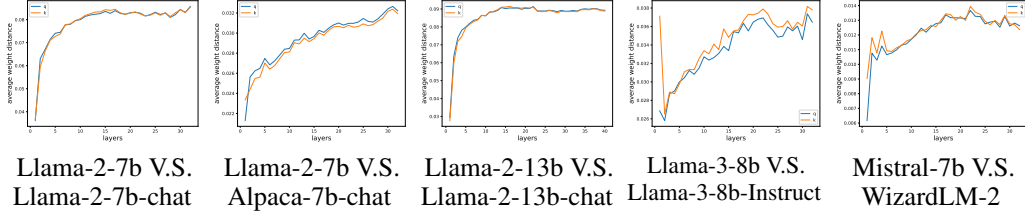

| Llama-2-7b V.S. Llama-2-7b-chat | Llama-2-7b V.S. Alpaca-7b-chat | Llama-2-13b V.S. Llama-2-13b-chat | Llama-3-8b V.S. Llama-3-8b-Instruct | Mistral-7b V.S. WizardLM-2 |

Figure 7: Average difference of weight vector pairs in the query and key across the layers of different LLMs (including Llama-2 [35], Alpaca [33], Llama-3 [35], Mistral [16], and WizardLM [38]). We show that the average difference increases after the first several layers and stays high, which is the result of the fact that only near orthogonal weight vector pairs are updated during fine-tuning.

## 4.1 Comparing Different Versions of the Same LLM

Generally, most LLMs are trained in a pretrain-finetune paradigm where the language model is firstly pre-trained on a large dataset with a pre-train task such as predicting the next token and then finetuned to accommodate better the needs for downstream tasks. Generally, there will be many different versions of the same LLM released such as the base version of the LLM after pre-training and the finetuned version *e.g.* the chat version. By comparing the parameters of two different versions of the same LLM, we could investigate how fine-tuning changes the parameters.

We present the results of comparison between the base model Mistral-7B-v0.1 [16] and its fine-tuned version WizardLM-2 [38] in Fig. 6. Each sub-figure contains three figures. With the scatter figure in the middle, we show the cosine similarity between weight vector pairs in the base model and the fine-tuned version, where the x-axis corresponds to the base model and the y-axis corresponds to the fine-tuned model. Generally, the fine-tuning barely changes the angle between weight vector pairs. It means that the weights responsible for processing basic syntactic information or high-level semantic information in the base model are still responsible for processing the corresponding basic syntactic information or high-level semantic information after fine-tuning. With the histogram figure on the top and left of each sub-figure, we further present the average $L_2$ distance between weight vector pairs in the base model and the fine-tuned model. For the figure on the top, the x-axis corresponds to the cosine similarity of the weight vector pairs in the base model and the y-axis corresponds to the weight distance. Similarly, for the figure on the left, the x-axis corresponds to weight distance, and the y-axis corresponds to cosine similarity results for the fine-tuned model.

Generally, we find that the weight distances are high for weight vector pairs with nearly zero cosine similarity. **It indicates that fine-tuning mainly changes the weights corresponding to high-level semantic information processing and the weights corresponding to low-level information such as syntax are merely changed.** Intuitively, it results from the fact that pre-trained LLMs are good enough at processing basic syntactic information and fine-tuning changes in how they process high-level semantic information for downstream tasks. We further show the average distance between weight vector pairs across the layers of different base and fine-tuned models in Fig. 7. As a result of the fact that only near orthogonal weight vector pairs are updated during fine-tuning, the average distance increases after the first several layers and stays large, which is aligned with the results in Fig. 3. As previous work [23] also found that the layers in the middle are the most updated part during fine-tuning, we provide a new perspective on further understanding the phenomenon. Beyond that, since we could effectively calculate the cosine similarity of weight vector pairs, it enables an efficient method to reduce the number of trainable parameters during fine-tuning. **Note that being barely changed during fine-tuning indicates non-orthogonal weight vector pairs are important in processing basic syntactic information, which also agrees with empirical results in [12].**

We have also conducted experiments on more LLMs and received similar results including Llama2 [35], Alpaca [33], and Llama3 [35]. Please refer to Appendix B for more results.

## 4.2 Reducing the Trainable Parameters on Query and Key

Our observation provides us with an efficient way to reduce the trainable parameters during training. According to the results in Sec. 4.1, the non-orthogonal weight vector pairs in the query and key weight matrix do not need to be updated during fine-tuning, therefore we can reduce the number of

Table 1: Results of Llama-2 fine-tuned on the query and value in attention module with LoRA [15] and AWM. We report the evaluation results on TruthfulQA [20], GSM8K [4], and HellaSwag [39]. We also report the threshold of AWM and the portion of the masked query weight vector pair.

| Model | TruthfulQA | | GSM8K | HellaSwag | Threshold $\tau$ | Fixed weight |
| | mc1 | mc2 | acc | acc | | vector pairs (%) |
|---|---|---|---|---|---|---|
| Llama-2-7b | 26.07 | 39.45 | 4.85 | 57.99 | - | - |
| + LoRA(r=8) | 33.41 | 49.48 | 10.92 | 58.81 | - | - |
| + LoRA(r=8) + AWM(Ours) | 34.27 | 50.37 | 11.30 | 58.79 | 0.01 | 77.20 |
| + LoRA(r=8) + AWM(Ours) | 34.27 | 49.98 | **11.90** | 58.79 | 0.005 | 85.36 |
| + LoRA(r=8) + AWM(Ours) | **35.00** | **50.97** | 10.69 | **58.91** | 0.001 | 92.12 |
| + LoRA(r=2) | 32.19 | 48.64 | 10.99 | 58.73 | - | - |
| + LoRA(r=2) + AWM(Ours) | 33.41 | 49.78 | 11.30 | 58.91 | 0.01 | 77.20 |
| + LoRA(r=2) + AWM(Ours) | 33.29 | 49.14 | **12.81** | 58.83 | 0.005 | 85.36 |
| + LoRA(r=2) + AWM(Ours) | **33.66** | **49.98** | 11.37 | **58.93** | 0.001 | 92.12 |
| Llama-2-13b | 23.75 | 33.41 | 19.33 | 60.89 | - | - |
| + LoRA(r=8) | 30.72 | 44.74 | 18.42 | 61.31 | - | - |
| + LoRA(r=8) + AWM(Ours) | 30.48 | 45.40 | 19.41 | **61.45** | 0.01 | 79.32 |
| + LoRA(r=8) + AWM(Ours) | 30.84 | 45.09 | 19.26 | 61.29 | 0.005 | 87.04 |
| + LoRA(r=8) + AWM(Ours) | **32.07** | **45.88** | **19.48** | 61.28 | 0.001 | 93.40 |

trainable parameters before the fine-tuning is conducted and only update the nearly orthogonal weight vector pairs during fine-tuning. Therefore, we propose the method, namely Angle-based Weight Masking (AWM), to reduce the number of parameters during fine-tuning. With a threshold set as $\tau$, we only update the weight vector pairs $\mathbf{w}_{\{q,k\}}^{2i-1}$ and $\mathbf{w}_{\{q,k\}}^{2i}$ with $|\cos\alpha| < \tau$ where $\alpha$ is the angle between $\mathbf{w}_{\{q,k\}}^{2i-1}$ and $\mathbf{w}_{\{q,k\}}^{2i}$. The detailed algorithm is in Alg. 1.

Since our method is orthogonal to many popular parameter efficient fine-tuning methods [15, 6] that reduce the computational cost of fine-tuning an LLM, in this section, we conduct our experiments with LoRA [15]. We finetune Llama2-7b and Llama2-13b [35] on Alpaca [33] using LoRA and AWM. The query and the value weight of each model are fine-tuned with LoRA on Alpaca for 3 epochs. For more details on hyperparameter settings and results, please refer to Appendix A.

In Table 1, we present the evaluation results of LoRA combined with our AWM. We evaluate the fine-tuned model on TruthfulQA [20], GSM8K [4], and Hellaswag [39]. TruthfulQA measures the tendency of the model to reproduce common falsehoods online. GSM8K is a test of diverse grade school math problems. Hellaswag is a test of commonsense inference. We use lm-eval [11] and follow the setting

---

**Algorithm 1** Angle-based Weight Masking

**Input:** Query and Key parameters $\Theta_{qk} = \{\mathbf{w}_{\{q,k\}}^i | i \in [0,d)\}$, other parameters $\Theta_O$, threshold $\tau$ and loss on fine-tuning dataset $\mathcal{L}(\cdot)$

$\Theta_{Tr} \leftarrow \Theta_O$
**for** $\mathbf{w}_{\{q,k\}}^{2i-1}, \mathbf{w}_{\{q,k\}}^{2i} \in \Theta_{qk}$ **do**
$\quad \cos\alpha \leftarrow \frac{\mathbf{w}_{\{q,k\}}^{2i\top}\mathbf{w}_{\{q,k\}}^{2i-1}}{\|\mathbf{w}_{\{q,k\}}^{2i}\|\|\mathbf{w}_{\{q,k\}}^{2i-1}\|}$
$\quad$ **if** $\cos\alpha < \tau$ **then**
$\quad\quad \Theta_{Tr} \leftarrow \Theta_{Tr} U\{\mathbf{w}_{\{q,k\}}^{2i-1}, \mathbf{w}_{\{q,k\}}^{2i}\}$
$\quad$ **end if**
**end for**
$\Theta_{Tr}^* \leftarrow \min_{\Theta_{Tr}} \mathcal{L}(\Theta_{Tr})$

---

in Open LLM Leaderboard[2] to evaluate LLMs on these datasets where models are evaluated in 0-shot setting on TruthfulQA, 5-shot on GSM8K, and 10-shot on Hellaswag. As shown in Table 1, our proposed method could largely reduce the trainable parameters in the query while maintaining or even boosting the performance. As shown in Table 2, we also conduct experiments following the setting in LoftQ [19] where we fine-tune Llama-2-7b, Mistral-7B, and Phi-2 on wikitext-2 and GSM8K. Generally, our proposed method improves the performance while reducing the number of trainable parameters. We conjecture that the performance is boosted because the orthogonal weight pairs in the query corresponding to basic syntactic information are fixed, which prevents the model from overfitting the basic syntactic information in the fine-tuning dataset. **Generally, the benefits of our**

Table 2: Results of LLMs fine-tuned on WikiText-2 [22] and GSM8K [4] with LoftQ [19] and AWM. Models are fine-tuned through causal language modelling on training sets and are tested on validation/test sets. We also report the threshold of AWM and the portion of the masked query weight vector pair.

| Model | WikiText-2 perplexity | GSM8K acc | Threshold $\tau$ | Fixed weight vector pairs (%) |
|---|---|---|---|---|
| Llama-2-7b + LoftQ | 5.518 | 39.20 | - | - |
| + LoftQ + AWM(Ours) | 5.485 | **40.86** | 0.01 | 77.20 |
| + LoftQ + AWM(Ours) | 5.483 | 38.44 | 0.005 | 85.36 |
| + LoftQ + AWM(Ours) | **5.480** | 38.06 | 0.001 | 92.12 |
| Mistral-7B + LoftQ | 6.423 | 54.51 | - | - |
| + LoftQ + AWM(Ours) | **6.335** | 55.12 | 0.01 | 19.72 |
| + LoftQ + AWM(Ours) | 6.340 | **55.88** | 0.005 | 21.56 |
| + LoftQ + AWM(Ours) | 6.337 | 55.80 | 0.001 | 23.04 |
| Phi-2 + LoftQ | 9.553 | 48.75 | - | - |
| + LoftQ + AWM(Ours) | 9.766 | 51.71 | 0.01 | 32.44 |
| + LoftQ + AWM(Ours) | 9.829 | 52.01 | 0.005 | 37.00 |
| + LoftQ + AWM(Ours) | 9.836 | **52.92** | 0.001 | 40.72 |

**proposed method are twofold: 1) further reduce the number of trainable parameters during fine-tuning, and 2) fix the orthogonal query and key weight vector pairs to prevent overfitting.**

## 5  Conclusion and Limitation Discussion

We provide a new perspective to study how LLMs with Rotary Position Embedding (RoPE) utilize the position information. We show that non-orthogonal query and key weight vector pairs in RoPE draw higher attention to certain relative positions regardless of the input, which may correspond to processing basic syntactic information while the orthogonal query and key weight vector pairs in RoPE are more flexible with the relative position, which may correspond to processing high-level semantic information. Our analysis and various empirical results at the layer level, attention head level, and neuron level are provided in Sec. 3. By comparing the pre-trained model and the fine-tuned model, we show that fine-tuning mainly updates near orthogonal weight vector pairs and further propose a method to reduce the number of trainable parameters during LLM fine-tuning in Sec. 4.

The limitation of the proposed method is that it only applies to the query or the key weights which may limit the number of reduced trainable parameters. Specifically, the query and key weight vector pairs are near orthogonal at random initialization, which makes the non-orthogonal query and key weight vector pairs non-trivial. The phenomena presented in previous works and in this paper imply that the non-orthogonal query and key weight vector pairs play an important role in LLMs using RoPE, which requires further investigation and we leave it for future works. This paper provides a new perspective on investigating large language models (LLMs), which have the potential to make LLMs more accessible and effective for various applications. However, we must also address the potential risks associated with their misuse.

## Footnotes

‡Corresponding author. This work was in part supported by NSFC (92370201, 62222607) and Shanghai Municipal Science and Technology Major Project under Grant 2021SHZDZX0102.

[1]The focused relative position is $\dfrac{\arctan(\frac{\|\mathbf{w}_q^{2i}\|}{\|\mathbf{w}_q^{2i-1}\|}) - \arctan(\frac{\|\mathbf{w}_k^{2i}\|}{\|\mathbf{w}_k^{2i-1}\|})}{\theta_i}$

[2]https://huggingface.co/spaces/HuggingFaceH4/open_llm_leaderboard

[3]https://huggingface.co/spaces/tloen/alpaca-lora

[4]https://huggingface.co/spaces/HuggingFaceH4/open_llm_leaderboard

## References

[1] Tom Brown, Benjamin Mann, Nick Ryder, Melanie Subbiah, Jared D Kaplan, Prafulla Dhariwal, Arvind Neelakantan, Pranav Shyam, Girish Sastry, Amanda Askell, et al. Language models are few-shot learners. *Advances in neural information processing systems*, 33:1877–1901, 2020.

[2] Aakanksha Chowdhery, Sharan Narang, Jacob Devlin, Maarten Bosma, Gaurav Mishra, Adam Roberts, Paul Barham, Hyung Won Chung, Charles Sutton, Sebastian Gehrmann, et al. Palm: Scaling language modeling with pathways. *Journal of Machine Learning Research*, 24(240):1–113, 2023.

[3] Yung-Sung Chuang, Yujia Xie, Hongyin Luo, Yoon Kim, James Glass, and Pengcheng He. Dola: Decoding by contrasting layers improves factuality in large language models. *arXiv preprint arXiv:2309.03883*, 2023.

[4] Karl Cobbe, Vineet Kosaraju, Mohammad Bavarian, Mark Chen, Heewoo Jun, Lukasz Kaiser, Matthias Plappert, Jerry Tworek, Jacob Hilton, Reiichiro Nakano, et al. Training verifiers to solve math word problems. *arXiv preprint arXiv:2110.14168*, 2021.

[5] Tim Dettmers, Mike Lewis, Younes Belkada, and Luke Zettlemoyer. Llm.int8(): 8-bit matrix multiplication for transformers at scale. *CoRR*, abs/2208.07339, 2022.

[6] Tim Dettmers, Artidoro Pagnoni, Ari Holtzman, and Luke Zettlemoyer. Qlora: Efficient finetuning of quantized llms. *Advances in Neural Information Processing Systems*, 36, 2024.

[7] Tim Dettmers and Luke Zettlemoyer. The case for 4-bit precision: k-bit inference scaling laws. In *International Conference on Machine Learning*, pages 7750–7774. PMLR, 2023.

[8] Jacob Devlin, Ming-Wei Chang, Kenton Lee, and Kristina Toutanova. BERT: pre-training of deep bidirectional transformers for language understanding. In Jill Burstein, Christy Doran, and Thamar Solorio, editors, *Proceedings of the 2019 Conference of the North American Chapter of the Association for Computational Linguistics: Human Language Technologies, NAACL-HLT 2019, Minneapolis, MN, USA, June 2-7, 2019, Volume 1 (Long and Short Papers)*, pages 4171–4186. Association for Computational Linguistics, 2019.

[9] Zhengxiao Du, Yujie Qian, Xiao Liu, Ming Ding, Jiezhong Qiu, Zhilin Yang, and Jie Tang. Glm: General language model pretraining with autoregressive blank infilling. *arXiv preprint arXiv:2103.10360*, 2021.

[10] Victoria Fromkin, Robert Rodman, and Nina Hyams. An lntroduction to language. *An lntroduction to Language*, 2014.

[11] Leo Gao, Jonathan Tow, Baber Abbasi, Stella Biderman, Sid Black, Anthony DiPofi, Charles Foster, Laurence Golding, Jeffrey Hsu, Alain Le Noac'h, Haonan Li, Kyle McDonell, Niklas Muennighoff, Chris Ociepa, Jason Phang, Laria Reynolds, Hailey Schoelkopf, Aviya Skowron, Lintang Sutawika, Eric Tang, Anish Thite, Ben Wang, Kevin Wang, and Andy Zou. A framework for few-shot language model evaluation, 12 2023.

[12] Andrey Gromov, Kushal Tirumala, Hassan Shapourian, Paolo Glorioso, and Daniel A. Roberts. The unreasonable ineffectiveness of the deeper layers. *CoRR*, abs/2403.17887, 2024.

[13] Song Han, Jeff Pool, John Tran, and William Dally. Learning both weights and connections for efficient neural network. *Advances in neural information processing systems*, 28, 2015.

[14] Neil Houlsby, Andrei Giurgiu, Stanislaw Jastrzebski, Bruna Morrone, Quentin De Laroussilhe, Andrea Gesmundo, Mona Attariyan, and Sylvain Gelly. Parameter-efficient transfer learning for nlp. In *International conference on machine learning*, pages 2790–2799. PMLR, 2019.

[15] Edward J. Hu, Yelong Shen, Phillip Wallis, Zeyuan Allen-Zhu, Yuanzhi Li, Shean Wang, Lu Wang, and Weizhu Chen. Lora: Low-rank adaptation of large language models. In *The Tenth International Conference on Learning Representations, ICLR 2022, Virtual Event, April 25-29, 2022*. OpenReview.net, 2022.

[16] Albert Q Jiang, Alexandre Sablayrolles, Arthur Mensch, Chris Bamford, Devendra Singh Chaplot, Diego de las Casas, Florian Bressand, Gianna Lengyel, Guillaume Lample, Lucile Saulnier, et al. Mistral 7b. *arXiv preprint arXiv:2310.06825*, 2023.

[17] Yann LeCun, John Denker, and Sara Solla. Optimal brain damage. *Advances in neural information processing systems*, 2, 1989.

[18] Xiang Lisa Li and Percy Liang. Prefix-tuning: Optimizing continuous prompts for generation. In Chengqing Zong, Fei Xia, Wenjie Li, and Roberto Navigli, editors, *Proceedings of the 59th Annual Meeting of the Association for Computational Linguistics and the 11th International Joint Conference on Natural Language Processing, ACL/IJCNLP 2021, (Volume 1: Long Papers), Virtual Event, August 1-6, 2021*, pages 4582–4597. Association for Computational Linguistics, 2021.

[19] Yixiao Li, Yifan Yu, Chen Liang, Nikos Karampatziakis, Pengcheng He, Weizhu Chen, and Tuo Zhao. Loftq: Lora-fine-tuning-aware quantization for large language models. In *The Twelfth International Conference on Learning Representations, ICLR 2024, Vienna, Austria, May 7-11, 2024*. OpenReview.net, 2024.

[20] Stephanie Lin, Jacob Hilton, and Owain Evans. Truthfulqa: Measuring how models mimic human falsehoods. *arXiv preprint arXiv:2109.07958*, 2021.

[21] Zichang Liu, Jue Wang, Tri Dao, Tianyi Zhou, Binhang Yuan, Zhao Song, Anshumali Shrivastava, Ce Zhang, Yuandong Tian, Christopher Re, et al. Deja vu: Contextual sparsity for efficient llms at inference time. In *International Conference on Machine Learning*, pages 22137–22176. PMLR, 2023.

[22] Stephen Merity, Caiming Xiong, James Bradbury, and Richard Socher. Pointer sentinel mixture models, 2016.

[23] Abhishek Panigrahi, Nikunj Saunshi, Haoyu Zhao, and Sanjeev Arora. Task-specific skill localization in fine-tuned language models. In *International Conference on Machine Learning*, pages 27011–27033. PMLR, 2023.

[24] Bowen Peng, Jeffrey Quesnelle, Honglu Fan, and Enrico Shippole. Yarn: Efficient context window extension of large language models. *arXiv preprint arXiv:2309.00071*, 2023.

[25] Jonas Pfeiffer, Aishwarya Kamath, Andreas Rücklé, Kyunghyun Cho, and Iryna Gurevych. Adapterfusion: Non-destructive task composition for transfer learning. *arXiv preprint arXiv:2005.00247*, 2020.

[26] Alec Radford, Jeffrey Wu, Rewon Child, David Luan, Dario Amodei, Ilya Sutskever, et al. Language models are unsupervised multitask learners. *OpenAI blog*, 1(8):9, 2019.

[27] Colin Raffel, Noam Shazeer, Adam Roberts, Katherine Lee, Sharan Narang, Michael Matena, Yanqi Zhou, Wei Li, and Peter J. Liu. Exploring the limits of transfer learning with a unified text-to-text transformer. *J. Mach. Learn. Res.*, 21:140:1–140:67, 2020.

[28] John Saeed. Semantics: The meaning of words and sentences. In *The Routledge Handbook of Linguistics*, pages 153–168. Routledge, 2015.

[29] Mohammad Samragh, Mehrdad Farajtabar, Sachin Mehta, Raviteja Vemulapalli, Fartash Faghri, Devang Naik, Oncel Tuzel, and Mohammad Rastegari. Weight subcloning: direct initialization of transformers using larger pretrained ones. *arXiv preprint arXiv:2312.09299*, 2023.

[30] Peter Shaw, Jakob Uszkoreit, and Ashish Vaswani. Self-attention with relative position representations. In Marilyn A. Walker, Heng Ji, and Amanda Stent, editors, *Proceedings of the 2018 Conference of the North American Chapter of the Association for Computational Linguistics: Human Language Technologies, NAACL-HLT, New Orleans, Louisiana, USA, June 1-6, 2018, Volume 2 (Short Papers)*, pages 464–468. Association for Computational Linguistics, 2018.

[31] Jianlin Su, Murtadha Ahmed, Yu Lu, Shengfeng Pan, Wen Bo, and Yunfeng Liu. Roformer: Enhanced transformer with rotary position embedding. *Neurocomputing*, 568:127063, 2024.

[32] Yutao Sun, Li Dong, Barun Patra, Shuming Ma, Shaohan Huang, Alon Benhaim, Vishrav Chaudhary, Xia Song, and Furu Wei. A length-extrapolatable transformer. *arXiv preprint arXiv:2212.10554*, 2022.

[33] Rohan Taori, Ishaan Gulrajani, Tianyi Zhang, Yann Dubois, Xuechen Li, Carlos Guestrin, Percy Liang, and Tatsunori B Hashimoto. Stanford alpaca: An instruction-following llama model, 2023.

[34] Ian Tenney, Dipanjan Das, and Ellie Pavlick. Bert rediscovers the classical nlp pipeline. *arXiv preprint arXiv:1905.05950*, 2019.

[35] Hugo Touvron, Thibaut Lavril, Gautier Izacard, Xavier Martinet, Marie-Anne Lachaux, Timothée Lacroix, Baptiste Rozière, Naman Goyal, Eric Hambro, Faisal Azhar, et al. Llama: Open and efficient foundation language models. *arXiv preprint arXiv:2302.13971*, 2023.

[36] Ashish Vaswani, Noam Shazeer, Niki Parmar, Jakob Uszkoreit, Llion Jones, Aidan N Gomez, Łukasz Kaiser, and Illia Polosukhin. Attention is all you need. *Advances in neural information processing systems*, 30, 2017.

[37] Elena Voita, Javier Ferrando, and Christoforos Nalmpantis. Neurons in large language models: Dead, n-gram, positional. *arXiv preprint arXiv:2309.04827*, 2023.

[38] Can Xu, Qingfeng Sun, Kai Zheng, Xiubo Geng, Pu Zhao, Jiazhan Feng, Chongyang Tao, and Daxin Jiang. Wizardlm: Empowering large language models to follow complex instructions. *arXiv preprint arXiv:2304.12244*, 2023.

[39] Rowan Zellers, Ari Holtzman, Yonatan Bisk, Ali Farhadi, and Yejin Choi. Hellaswag: Can a machine really finish your sentence? *arXiv preprint arXiv:1905.07830*, 2019.

[40] Shaofeng Zhang, Jinfa Huang, Qiang Zhou, Zhibin Wang, Fan Wang, Jiebo Luo, and Junchi Yan. Continuous-multiple image outpainting in one-step via positional query and a diffusion-based approach. 2024.

# A  Details of Experiments

## A.1  Details of AWM

As shown in Alg. 1, with a threshold $\tau$, we fix the weight vector pairs with absolute cosine similarity larger than the threshold $\tau$. We then fine-tune the model with LoRA on Alpaca. We set the hyper-parameters following alpaca-lora [3]. We fine-tune the query and the value with Lora alpha at $16$ and rank at $8$ or $2$. The learning rate is $3e - 4$, batch size is $128$, cutoff length is $512$, and the dropout for LoRA is $0.05$. Since the absolute cosine similarity is large in the first 3 layers, we did not apply our method to the first 3 layers of Llama-2-7b and Llama-2-13b.

For evaluation, we follow open llm leaderboard [4] to evaluate the finetuned model on Hellaswag, TruthfulQA and GSM8K. All the hyper-parameter settings are the same as those on the open LLM leaderboard. All the experiments are conducted on one NVIDIA GeForce RTX 4090.

## A.2  Details of Attention Visualization

We take two simple questions as input. We record the query and key of each layer and calculate the attention score of different attention heads. We present the attention head with the largest and the smallest average absolute cosine similarity between weight vector pairs of the first layer of Llama-2-7b-chat and Mistral 7B Instruct v0.2.

## A.3  Details of Model Comparison

We compare various models, including Llama-2-7b-chat, Llama-2-7 b, Llama-2-7 b finetuned on Alpaca, Mistral 7 B v0.1, WizzardLM2, Llama-3-8 B, and Llama-3-8B-Instruct. The phenomenon between these models is similar, with changes mainly on the near orthogonal weight vector pairs.

# B  Additional Experiemental Results

## B.1  Model Comparison

In this section, we provide more comparisons of the results of two versions of the same fine-tuned model.

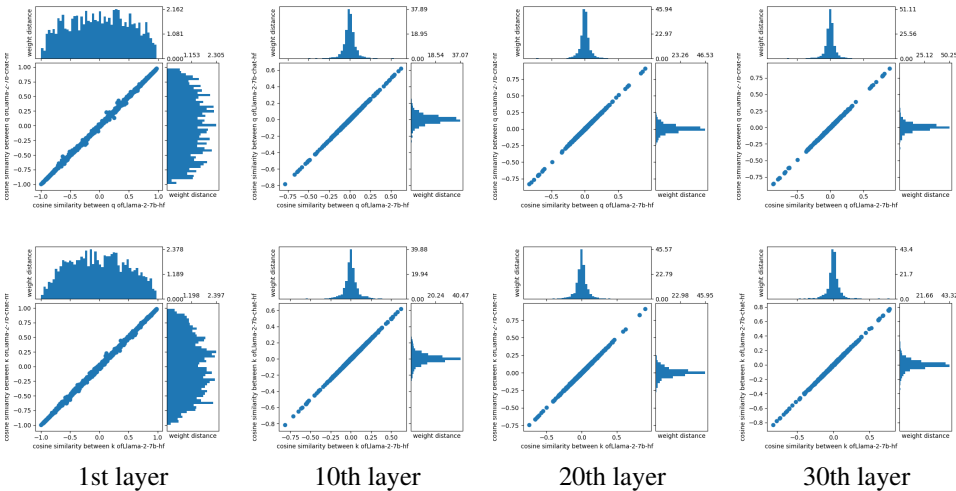

| 1st layer | 10th layer | 20th layer | 30th layer |

Figure 8: Comparsion Results between Llama-2-7b and Llama-2-7b-chat. Similar to Fig. 6.

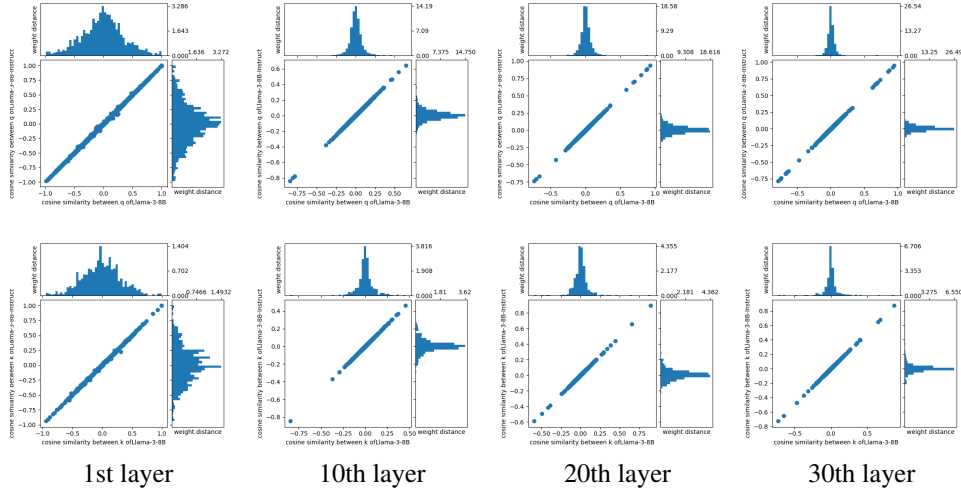

| 1st layer | 10th layer | 20th layer | 30th layer |

Figure 9: Comparsion Results between Llama-3-8B and Llama-3-8B-Instruct. Similar to Fig. 6.

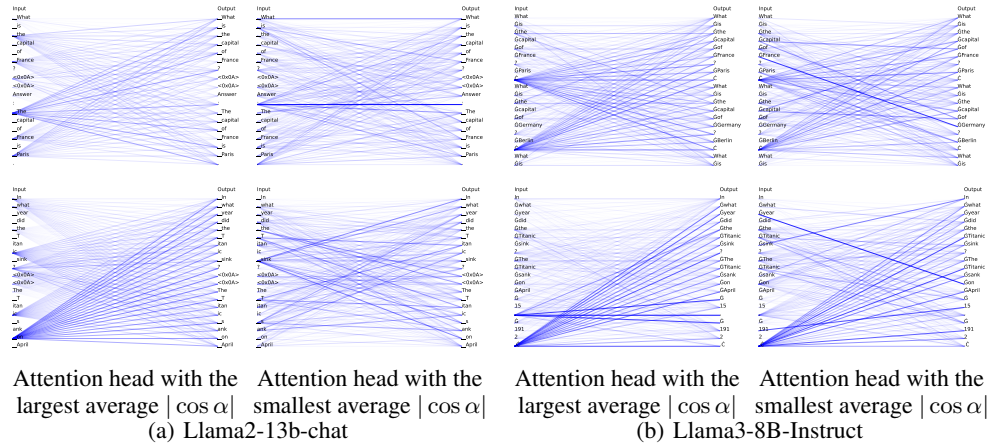

Attention head with the largest average $|\cos\alpha|$ | Attention head with the smallest average $|\cos\alpha|$ | Attention head with the largest average $|\cos\alpha|$ | Attention head with the smallest average $|\cos\alpha|$
(a) Llama2-13b-chat
(b) Llama3-8B-Instruct

Figure 10: Attention visualization of the attention heads with the largest or the smallest average absolute cosine similarity $|\cos\alpha|$ across the weight vector pairs in RoPE, where the lower transparency means the higher attention score. We demonstrate the results for the first layer of Llama2 13b chat and Llama3 8B Instruct. Note that Llama3 8B have a much lower $|\cos\alpha|$ (0.46 max and 0.17 min) comparing to Llama2 (0.70 max and 0.31 min).

## B.2 Attention Visualization

We provide more attention visualization results on Llama-2-13b and Llama3 8B in Fig. 10. Note that the attention head at the first layer of Llama3 8B has a relatively small average absolute cosine similarity compared to Llama2 7b and Llama2 13b.

## B.3 Pair-wise Cosine Similarity

In this section, we further provide the visualization of cosine similarity between the weight vector pairs in different Mistral 7B v0.1 layers in Fig. 11. In the future, we hope to extend our method and apply it to further models such as multi-modal LLMs or position information related to image processing [40].

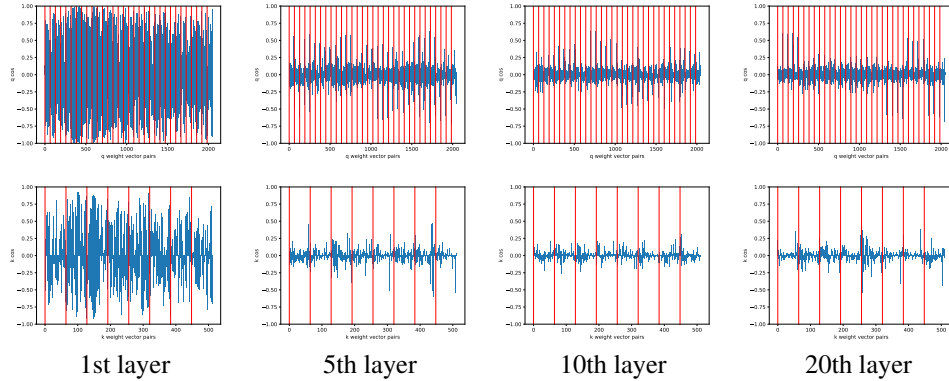

| 1st layer | 5th layer | 10th layer | 20th layer |

Figure 11: We show the cosine similarity of each weight vector pair in the query and the key of different layers of the Mistral 7B v0.1. Each column corresponds to different layers. The top row is the results for the query, and the bottom row is the results for the key. The attention heads are separated with vertical red lines. We show that even within one head, the cosine similarity of different weight vector pairs differs.

